# CCD Neural Network Processors for Pattern Recognition

Alice M. Chiang        Michael L. Chuang        Jeffrey R. LaFranchise

MIT Lincoln Laboratory
244 Wood Street
Lexington, MA 02173

## Abstract

A CCD-based processor that we call the NNC2 is presented. The NNC2 implements a fully connected 192-input, 32-output two-layer network and can be cascaded to form multilayer networks or used in parallel for additional input or output nodes. The device computes $1.92 \times 10^9$ connections/sec when clocked at 10 MHz. Network weights can be specified to six bits of accuracy and are stored on-chip in programmable digital memories. A neural network pattern recognition system using NNC2 and CCD image feature extractor (IFE) devices is described. Additionally, we report a CCD output circuit that exploits inherent nonlinearities in the charge injection process to realize an adjustable-threshold sigmoid in a chip area of $40 \times 80 \ \mu m^2$.

## 1  INTRODUCTION

A neural network chip based on charge-coupled device (CCD) technology, the NNC2, is presented. The NNC2 implements a fully connected two-layer net and can be cascaded to form multilayer networks. An image feature extractor (IFE) device (Chiang and Chuang, 1991) is briefly reviewed. The IFE is suited for neural networks with local connections and shared weights and can also be used for image preprocessing tasks. A neural network pattern recognition system based on feature extraction using IFEs and classification using NNC2s is proposed. The efficacy of neural networks with local connections and shared weights for feature extraction in character

recognition and phoneme recognition tasks has been demonstrated by researchers such as (LeCun *et. al.* 1989) and (Waibel *et. al.*, 1989), respectively. More complex recognition tasks are likely to prove amenable to a system using locally connected networks as a front end with outputs generated by a highly-connected classifier. Both the IFE and the NNC2 are hybrids composed of analog and digital components. Network weights are stored digitally while neuron states and computation results are represented in analog form. Data enter and leave the devices in digital form for ease of integration into digital systems.

The sigmoid is used in many network models as the nonlinear neuron output function. We have designed, fabricated and tested a compact CCD sigmoidal output circuit that is described below. The paper concludes with a discussion of strategies for implementing networks with particularly high or low fan-in to fan-out ratios.

## 2    THE NNC2 AND IFE DEVICES

The NNC2 is a neural network processor that implements a fully connected two-layer net with 192 input nodes and 32 output nodes. The device is an expanded version of a previous neural network classifier (NNC) chip (Chiang, 1990) hence the appellation "NNC2." The NNC2 consists of a 192-stage CCD tapped delay line for holding and shifting input values, 192 four-quadrant multipliers, and 192 32-word local memories for weight storage. When clocked at 10 MHz, the NNC2 performs $1.92 \times 10^9$ connections/sec. The device was fabricated using a 2-$\mu$m minimum feature size double-metal, double-polysilicon CCD/CMOS process. The NNC2 measures $8.8 \times 9.2$ mm$^2$ and is depicted in Figure 1.

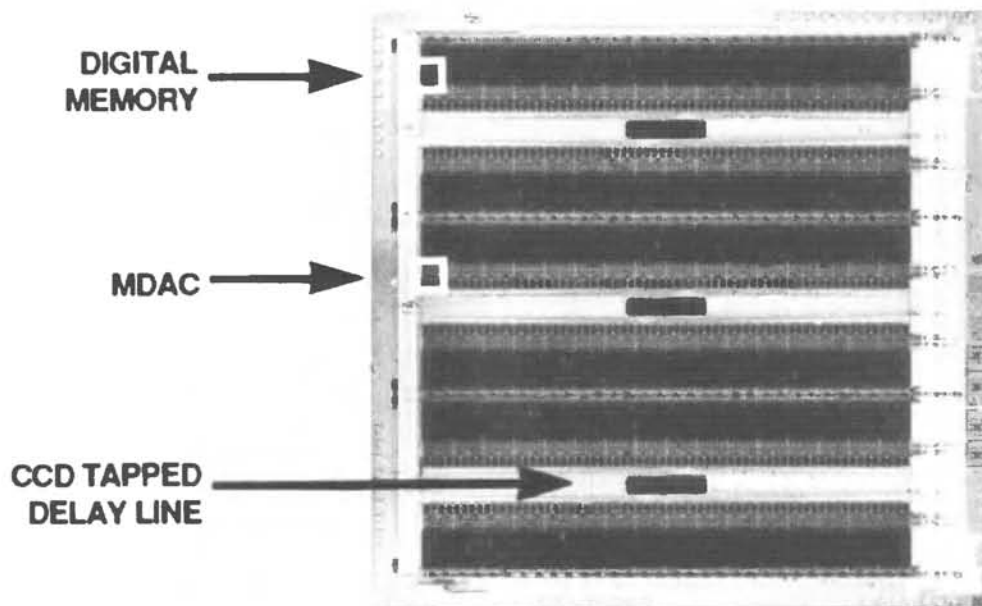

DIGITAL
MEMORY

MDAC

CCD TAPPED
DELAY LINE

Figure 1: Photomicrograph of the NNC2

Tests indicate that the NNC2 has an output dynamic range exceeding 42 dB. Figure 2 shows the output of the NNC2 when the input consists of the cosine waveforms $f_n = 0.2cos(2\pi 2n/192) + 0.4cos(2\pi 3n/192)$ and the weights are set to

$cos(2\pi nk/192)$, $k = \pm 1, \pm 2, ..., \pm 16$. Due to the orthogonality of sinusoids of different frequencies, the output correlations $g_k = \sum_{n=0}^{191} f_n cos(2\pi nk/192)$ should yield scaled impulses with amplitudes of $\pm 0.2$ and $\pm 0.4$ for $k = \pm 2$ and $\pm 3$ only; this is indeed the case as the output (lower trace) in Figure 2 shows. This test demonstrates the linearity of the weighted sum (inner product) computed by the NNC2.

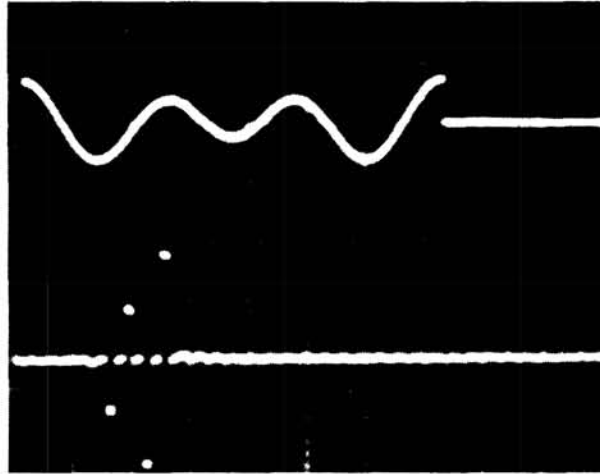

Figure 2: Response of the NNC2 to input cosine waveforms

Locally connected, shared weight networks can be implemented using the IFE which raster scans up to 20 sets of 7×7 weights over an input image. At every window position the inner product of the windowed pixels and each of the 20 sets of weights is computed. For additonal details, see (Chiang and Chuang, 1991). The IFE and the NNC2 share a number of common features that are described below.

## 2.1  MDACS

The multiplications of the inner product are performed in parallel by multiplying-D/A-converters (MDACs), of which there are 192 in the NNC2 and 49 in the IFE. Each MDAC produces a charge packet proportional to the product of an input and a digital weight. The partial products are summed on an output line common to all the MDACs, yielding a complete inner product every clock cycle. The design and operation of an MDAC are described in detail in (Chiang, 1990). Using a 2-$\mu$m design rule, a four-quadrant MDAC with 8-bit weights occupies an area of 200×200 $\mu$m$^2$.

## 2.2  WEIGHT STORAGE

The NNC2 and IFE feature on-chip digital storage of programmable network weights, specified to 6 and 8 bits, respectively. The NNC2 contains 192 local memories of 32 words each, while the IFE has forty-nine 20-word memories. Individual words can be addressed by means of a row pointer and a column pointer. Each bit of the CCD shift register memories is equipped with a feedback enable switch that obviates the need to refresh the volatile CCD storage medium explictly; words are

rewritten as they are read for use in computation, so that no cycles need be devoted to memory refresh.

## 2.3   INPUT BUFFER

Inputs to the NNC2 are held in a 192-stage CCD analog floating-gate tapped delay line. At each stage the floating gate is coupled to the input of the corresponding MDAC, permitting inputs to be sensed nondestructively for computation. The NNC2 delay line is composed of three 64-stage subsections (see Figure 1). This partionning allows the NNC2 to compute either the weighted sum of 192 inputs or three 64-point inner products. The latter capability is well-matched to Time-Delay Neural Networks (TDNNs) that implement a moving temporal window for phoneme recognition (Waibel *et. al.*, 1989). The IFE contains a similar 775-stage delay line that holds six lines of a 128-pixel input image plus an additional seven pixels. Taps are placed on the first seven of every 128 stages in the IFE delay line so that the 1-dimensional line emulates a 2-dimensional window.

## 3   CCD SIGMOIDAL OUTPUT CIRCUIT

A sigmoidal charge-domain nonlinear detection circuit is shown in Figure 3. The circuit has a programmable input-threshold controlled by the amplitude of the transfer gate voltage, $V_{TG}$. If the incoming signal charge is below the threshold set by $V_{TG}$ no charge is transferred to the output port and the incoming signal is ignored. If the input is above threshold, the amount of charge transferred to the output port is the difference between the charge input and the threshold level. The circuit design is based on the ability to calculate the charge transfer efficiency from an $n^+$ diffusion region over a bias gate to a receiving well as a function of device parameters and exploits the fact that under certain operating conditions a nonlinear dependence exists between the input and output charge (Thornber, 1971). The maximum output produced can be bounded by the size and gate voltage of the receiving well. The predicted and measured responses of the circuit for two different threshold levels are shown in the bottom of Figure 3. The circuit has an area of $40 \times 80$ $\mu$m$^2$ and can be integrated with the NNC2 or IFE chips to perform both the weighted-sum and output-nonlinearity computations on a single device.

## 4   DESIGN STRATEGIES

The NNC2 uses a time-multiplexed output (TMO) structure (Figure 4a), where the number of multipliers and the number of local memories is equal to the number of inputs, $N$. The depth of each local memory is equal to the number of output nodes, $M$, and the outputs are computed serially as each set of weights is read in sequence from the memories. A 256-input, 256-output device with 64k 8-bit weights has been designed and can be realized in a chip area of $14 \times 14$ mm$^2$. This chip is reconfigurable so that a single such device can be used to implement multilayer networks. If a network with a large (>1000) number of input nodes is required, then a time-multiplexed input (TMI) architecture with $M$ multipliers may be more suitable (Figure 4b). In contrast to a TMO system that computes the $M$ inner products

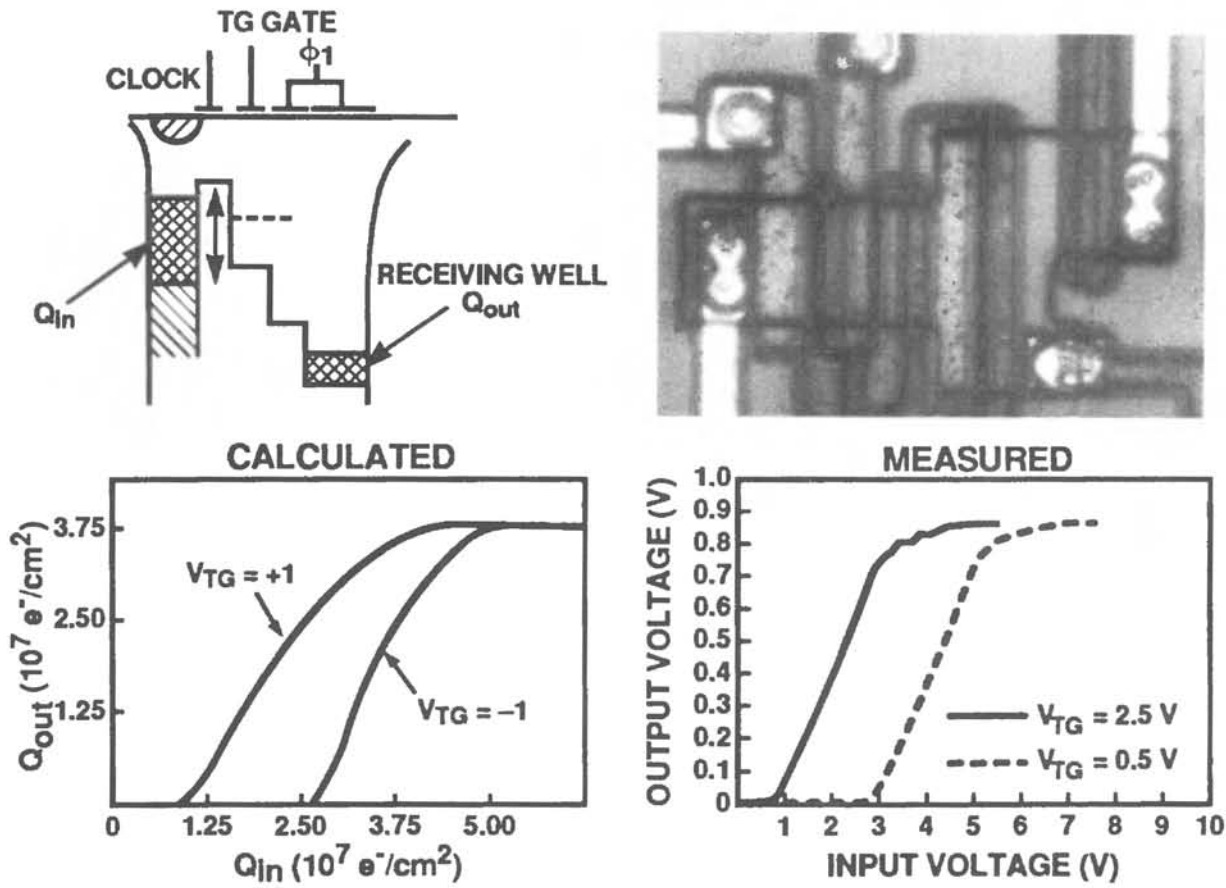

Figure 3: Schematic, micrograph, and test results of the sigmoid circuit

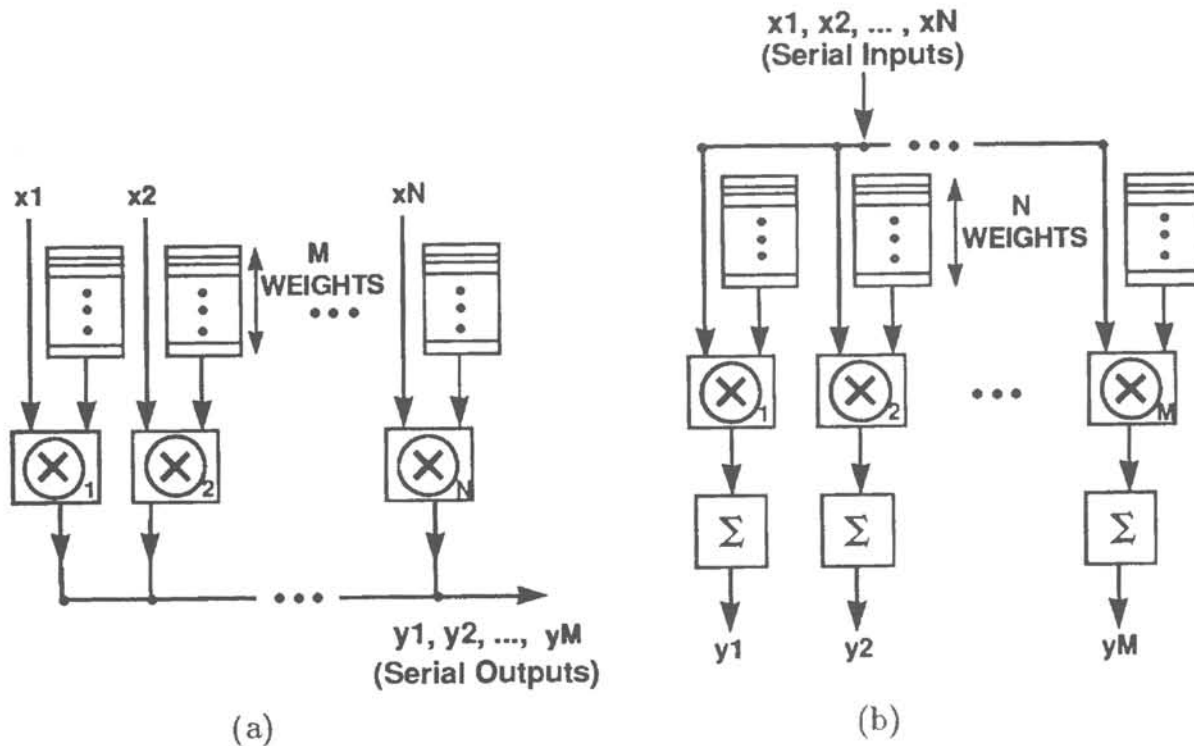

Figure 4: (a) Time-multiplexed output (TMO), (b) Time-multiplexed input (TMI)

sequentially (the multiplications of each inner product are performed in parallel), a TMI structure performs $N$ sets of $M$ multiplications each (all $M$ inner products are serially computed in parallel). As each input element arrives it is broadcast to all $M$ multipliers. Each multiplier multiplies the input by an appropriate weight from its $N$-word deep local memory and places the result in an accumulator. The $M$ inner products appear in the accumulators one cycle after receipt of the final, $N^{th}$ input.

## 5   SUMMARY

We have presented the NNC2, a CCD chip that implements a fully connected two-layer network at the rate of $1.92 \times 10^9$ connections/second. The NNC2 may be used in concert with IFE devices to form a CCD-based neural network pattern recogniton system or as a co-processor to speed up neural network simulations on conventional computers. A VME-bus board for the NNC2 is presently being constructed. A compact CCD circuit that generates a sigmoidal output function was described, and finally, the relative merits of time-multiplexing input or output nodes in neural network devices were enumerated. Table 1 below is a comparison of recent neural network chips.

| | MIT LINCOLN LAB NNC2 | CIT NN | INTEL ETANN | MITSUBISHI NN | AT&T NN | HITACHI WSINN | ADAPT. SOL. X1 |
|---|---|---|---|---|---|---|---|
| No. OF OUTPUT NODES | 32 | 256 | TWO 64 | 168 | 16 (or 256) | 576 | 64 |
| No. OF INPUT NODES | 192 | 256 | TWO 64 | 168 | 256 (or 16) | 64 | 4 k |
| SYNAPSE ACCURACY | 6 b · ANALOG | 1 b · ANALOG | ANALOG · ANALOG | ANALOG · ANALOG | 3 b · 6 b | 8 b · 9 b | 9 b · 16 b |
| PROGRAMMABLE SYNAPSES | 6 k | 64 k | 10 k | 28 k | 4 k | 37 k | 256 k |
| THROUGHPUT RATE ($10^9$ Connections/s) | 1.92 | 0.5 | 2 | ? | 5.1 | 1.2 | 1.6 |
| CHIP AREA (mm$^2$) | 8.8 · 9.2 | ? | 11.2 · 7.5 | 14.5 · 14.5 | 4.5 · 7 | 125 · 125 | 26.2 · 27.5 |
| CLOCK RATE | 10 MHz | 1.5 MHz | 400 kHz | ? | 20 MHz | 2.1 MHz[a] | 25 MHz |
| WEIGHT STORAGE | DIGITAL[b] | ANALOG | ANALOG | ANALOG | ANALOG | DIGITAL | DIGITAL |
| ON CHIP LEARNING | NO | NO | NO | YES[c] | NO | NO | YES |
| DESIGN RULE | 2 μm CCD/CMOS | 2 μm CCD | 1 μm CMOS | 1 μm CMOS | 0.9 μm CMOS | 0.8 μm CMOS | 0.8 μm CMOS |
| REPORTED AT: | NIPS 91 | IJCNN 90 | IJCNN 89 | ISSCC 91 | ISSCC 91 | IJCNN 90 | ISSCC 91 |

**NOTE:**
- a - CLOCK RATE FOR WSINN IS EXTRAPOLATED BASED ON 1/STEP TIME.
- b - NO DEGRADATION OBSERVED ON DIGITALLY STORED AND REFRESHED WEIGHTS.
- c - A SIMPLIFIED BOLTZMANN MACHINE LEARNING ALGORITHM IS USED.

Table 1: Selected neural network chips

### Acknowledgements

This work was supported by DARPA, the Office of Naval Research, and the Department of the Air Force. The IFE and NNC2 were fabricated by Orbit Semiconductor.

# References

A. J. Agranat, C. F. Neugebauer and A. Yariv, "A CCD Based Neural Network Integrated Circuit with 64k Analog Programmable Synapses," *IJCNN, 1990 Proceedings*, pp. II-551-II-555.

Y. Arima *et. al.*, "A 336-Neuron 28-k Synapse Self-Learning Neural Network Chip with branch-Neuron-Unit Architecture," in *ISSCC Dig. of Tech. Papers*, pp. 182-183, Feb. 1991.

B. E. Boser and E. Säckinger, "An Analog Neural Network Processor with Programmable Network Topology," in *ISSCC Dig. of Tech. Papers*, pp. 184-185, Feb. 1991.

A. M. Chiang, "A CCD Programmable Signal Processor," *IEEE Jour. Solid-State Circ.*, vol. 25, no. 6, pp. 1510-1517, Dec. 1990.

A. M. Chiang and M. L. Chuang, "A CCD Programmable Image Processor and its Neural Network Applications," *IEEE Jour. Solid-State Circ.*, vol. 26, no. 12, pp. 1894-1901, Dec. 1991.

D. Hammerstrom, "A VLSI Architecture for High-Performance, Low-Cost On-chip Learning," *IJCNN, 1990 Proceedings*, pp. II-537-II-543.

M. Holler *et. al.*, "An Electrically Trainable Artificial Neural Network (ETANN) with 10240 "Floating Gate" Synapses," *IJCNN, 1989 Proceedings*, pp. II-191-II-196.

Y. Le Cun *et. al.*, "Handwritten Digit Recognition with a Back-Propagation Network," in D. S. Touretzky (ed.), *Advances in Neural Information Processing Systems 2*, pp. 396-404, San Mateo, CA: Morgan Kaufmann, 1989.

K. K. Thornber, "Incomplete Charge Transfer in IGFET Bucket-Brigade Shift Registers," *IEEE Trans. Elect. Dev.*, vol. ED-18, no. 10, pp.941-950, 1971.

A. Waibel *et. al.*, "Phoneme Recognition Using Time-Delay Neural Networks," *IEEE Trans. on Acoust., Speech, Sig. Proc.*, vol. 37, no. 3, pp. 329-339, March 1989.

M. Yasunaga *et. al.*, "Design, Fabrication and Evaluation of a 5-Inch Wafer Scale Neural Network LSI Composed of 576 Digital Neurons," *IJCNN, 1990 Proceedings*, pp. II-527-II-535.
